# NEURAL NETWORK STAR PATTERN RECOGNITION FOR SPACECRAFT ATTITUDE DETERMINATION AND CONTROL

Phillip Alvelda,  A. Miguel San Martin
The Jet Propulsion Laboratory,
California Institute of Technology,
Pasadena, Ca. 91109

## ABSTRACT

Currently, the most complex spacecraft attitude determination and control tasks are ultimately governed by ground-based systems and personnel. Conventional on-board systems face severe computational bottlenecks introduced by serial microprocessors operating on inherently parallel problems. New computer architectures based on the anatomy of the human brain seem to promise high speed and fault-tolerant solutions to the limitations of serial processing. This paper discusses the latest applications of artificial neural networks to the problem of star pattern recognition for spacecraft attitude determination.

## INTRODUCTION

By design, a conventional on-board microprocessor can perform only one comparison or calculation at a time. Image or pattern recognition problems involving large template sets and high resolution can require an astronomical number of comparisons to a given database. Typical mission planning and optimization tasks require calculations involving a multitude of parameters, where each element has an inherent degree of importance, reliability and noise. Even the most advanced supercomputers running the latest software can require seconds and even minutes to execute a complex pattern recognition or expert system task, often providing incorrect or inefficient solutions to problems that prove trivial to ground control specialists.

The intent of ongoing research is to develop a neural network based satellite attitude determination system prototype capable of determining its current three-axis inertial orientation. Such a system that can determine in real-time, which direction the satellite is facing, is needed in order to aim antennas, science instruments, and navigational equipment. For a satellite to be autonomous (an important criterion in interplanetary missions, and most particularly so in the event of a system failure), this task must be performed in a reasonable amount of time with all due consideration to actual environmental, noise and precision constraints.

## CELESTIAL ATTITUDE DETERMINATION

Under normal operating conditions there is a whole repertoire of spacecraft systems that operate in conjunction to perform the attitude determination task, the backbone of which is the Gyro. But a Gyro measures only changes in orientation. The current attitude is stored in

volatile on-board memory and is updated as the gyro system integrates velocity to provide change in angular position. When there is a power system failure for any reason such as a single-event-upset due to cosmic radiation, all currently stored attitude information is LOST!

One very attractive way of recovering attitude information with no a priori knowledge is by using on-board imaging and computer systems to:

1.) Image a portion of the sky,

2.) Compare the characteristic pattern of stars in the sensor field-of-view to an on-board star catalog,

3.) Thereby identify the stars in the sensor FOV [Field Of View],

4.) Retrieve the identified star coordinates,

5.) Transform and correlate FOV and real-sky coordinates to determine spacecraft attitude.

But the problem of matching a limited field of view that contains a small number of stars (out of billions and billions of them), to an on-board full-sky catalog containing perhaps thousands of stars has long been a severe computational bottleneck.

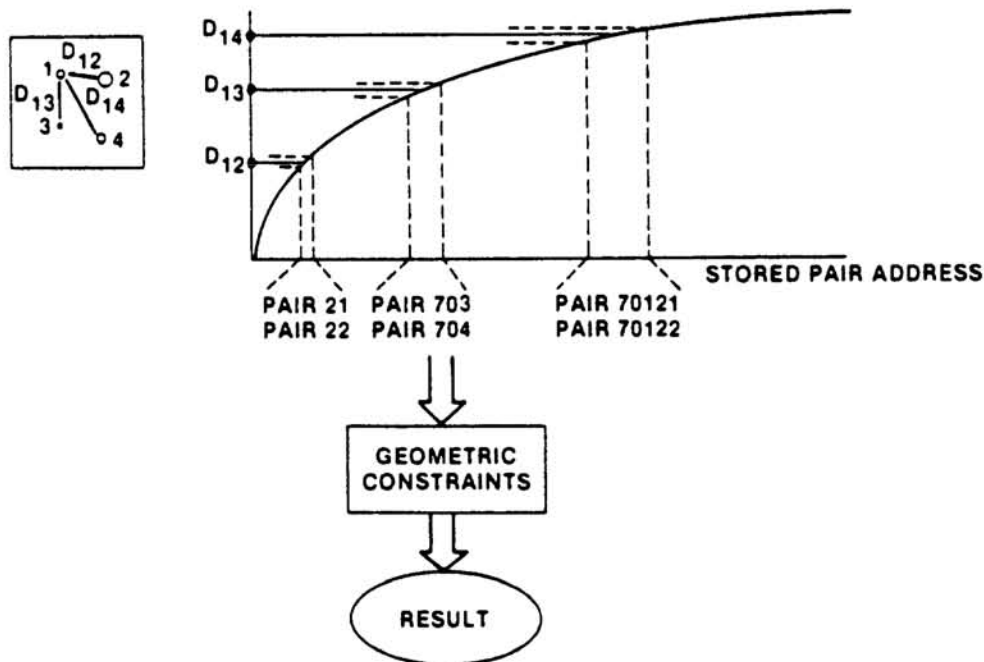

Figure 1.) Serial star I.D. catalog format and methodology.

The latest serial algorithm to perform this task requires approximately 650 KBytes of RAM to store the on-board star catalog. It incorporates a highly optimized algorithm which uses a motorola 68000 to search a sorted database of more than 70,000 star-pair distance values for correlations with the decomposed star pattern in the sensor FOV. It performs the identification process on the order of 1 second

with a success rate of 99 percent. **But it does not fit in the spacecraft on-board memory,** and therefore, no such system has flown on a planetary spacecraft.

- ● USES SUN SENSOR AND ATTITUDE MANEUVERS

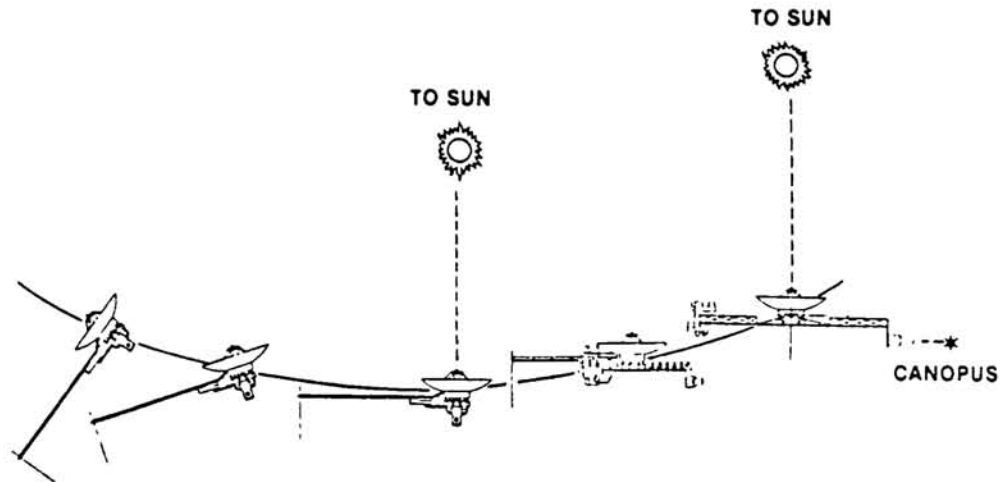

**Figure 2.)** Current Spacecraft attitude information recovery sequence.

As a result, state-of-the-art interplanetary spacecraft use several independent sensor systems in conjunction to determine attitude with no a priori knowledge. First, the craft is commanded to slew until a Sun Sensor (aligned with the spacecraft's major axis) has locked-on to the sun. The craft must then rotate around that axis until an appropriate star pattern at approximately ninety degrees to the sun is acquired to provide three-axis orientation information. The entire attitude acquisition sequence requires an absolute minimum of thirty minutes, and presupposes that all spacecraft actuator and maneuvering systems are operational. At the phenomenal rendezvous speeds involved in interplanetary navigation, a system failure near mission culmination could mean an almost complete loss of the most valuable scientific data while the spacecraft performs its initial attitude acquisition sequence.

## NEURAL MOTIVATION

The parallel architecture and collective computation properties of a neural network based system address several problems associated with the implementation and performance of the serial star ID algorithm. Instead of searching a lengthy database one element at a time, each stored star pattern is correlated with the field of view concurrently. And whereas standard memory storage technology requires one address in RAM per star-pair distance, the neural star pattern representations are stored in characteristic matrices of interconnections between neurons. This distributed data set representation has several desirable properties. First of all, the 2N redundancy of the serial star-pair scheme (i.e. which star is at which end of a pair) is discarded and a new more compressed representation emerges from the neuromorphic architecture. Secondly, noise, both statistical (i.e thermal noise) and systematic (i.e. sensor precision limitations), and pattern invariance characteristics are

incorporated directly into the preprocessing and neural architecture without extra circuitry.

### The first neural approach

The primary motivation from the NASA perspective is to improve satellite attitude determination performance and enable on-board system implementations. The problem methodology for the neural architecture is then slightly different than that of the serial model.

Instead of identifying every detected star in the field of view, the neural system identifies a single 'Guide Star' with respect to the pattern of dimmer stars around it, and correlates that star's known position with the sensor FOV to determine the pointing axis. If needed, only one other star is then required to fix the roll angle about that axis. So the core of the celestial attitude determination problem changes from multiple star identification and correlation, single star **pattern** identification.

The entire system consists of several modules in a marriage of different technologies. The first neural system architecture uses already mature(i.e. sensor/preprocessor) technologies where they perform well, and neural technology only where conventional systems prove intractable. With an eye towards rapid prototyping and implementation, the system was designed with technologies (such as neural VLSI) that will be available in less than one year.

## SYSTEM ARCHITECTURE

### The Star Tracker sensor system
The system input is based on the ASTROS II star tracker under development in the Guidance and Control section at the Jet Propulsion Laboratory. The Star tracker optical system images a defocussed portion of the sky (a star sub-field) onto a charged coupled device (C.C.D.). The tracker electronics then generate star centroid position and intensity information and passes this list to the preprocessing system.

### The Preprocessing system
This centroid and intensity information is passed to the preprocessing subsystem where the star pattern is treated to extract noise and pattern invariance. A 'pattern field-of-view' is defined as centered around the brightest (i.e. 'Guide Star') in the central portion of the sensor field-of-view. Since the pattern FOV radius is one half that of the sensor FOV the pattern for that 'Guide Star' is then based on a portion of the image that is complete, or invariant, under translational perturbation. The preprocessor then introduces rotational invariance to the 'guide-star' pattern by using only the distances of all other dimmer stars inside the pattern FOV to the central guide star.

These distances are then mapped by the preprocessor onto a two dimensional coordinate system of distance versus relative magnitude (normalized to the guide star, the brightest star in the Pattern FOV) to be sampled by the neural associative star catalog. The motivation for this distance map format become clear when issues involving noise invariance and memory capacity are considered.

Because the ASTROS Star Tracker is a limited precision instrument, most particularly in the absolute and relative intensity measures, two major problems arise. First, dimmer stars with intensities near the bottom of the dynamic range of the C.C.D. may or may not be included in the star pattern. So, the entire distance map is scaled to the brightest star such that the bright, high-confidence measurements are weighted more heavily, while the dimmer and possibly transient stars are of less importance to a given pattern. Secondly, since there are a very large number of stars in the sky, the uniqueness of a given star pattern is governed mostly by the relative star distance measures (which, by the way, are the highest precision measurements provided by the star tracker).

In addition, because of the limitations in expected neural hardware, a discrete number of neurons must sample a continuous function. To retain the maximum sample precision with a minimum number of neurons, the neural system uses the biological mechanism of a receptive field for hyperacuity. In other words, a number of neurons respond to a single distance stimulus. The process is analogous to that used on the defocussed image of a point source on the C.C.D. which was integrated over several pixels to generate a centroid at sub-pixel accuracies. To relax the demands on hardware development for the neural module, this point smoothing was performed in the preprocessor instead of being introduced into the neural network architecture and dynamics. The equivalent neural response function then becomes:

$$x_i = \sum_{k=1}^{N} \Psi_k \, e^{-(\mu_i - \mu_k)^2/\Delta}$$

where:

$x_i$    is the sampling activity of neuron i

$N$    is the number of stars in the Pattern Field Of View

$\mu_i$    is the position of neuron i on the sample axis

$\mu_k$ ·   is the position of the stimulus from star k on the sample axis

$\Psi_k$    is the magnitude scale factor of star k, normalized to the brightest star in the PFOV, the 'Guide star'

$\Delta$    is the width of the gaussian point spread function

**The Neural system**
The neural system, a 106 neuron, three-layer, feed-forward network, samples the scaled and smoothed distance map, to provide an output vector with the highest neural output activity representing the best match to one of the pre-trained guide star patterns. The network training algorithm uses the standard backwards error propagation

algorithm to set network interconnect weights from a training set of 'Guide Star' patterns derived from the software simulated sky and sensor models.

### Simulation testbed

The computer simulation testbed includes a realistic celestial field model, as well as a detector model that properly represents achievable position and intensity resolution, sensor scan rates, dynamic range, and signal to noise properties. Rapid identification of star patterns was observed in limited training sets as the simulated tracker was oriented randomly within the celestial sphere.

## PERFORMANCE RESULTS AND PROJECTIONS

In terms of improved performance the neural system was quite a success, but not however in the areas which were initially expected. While a VLSI implementation might yield considerable system speed-up, the digital simulation testbed neural processing time was of the same order as the serial algorithm, perhaps slightly better. The success rate of the serial system was already better than 99%. The neural net system achieved an accuracy of 100% when the systematic noise (i.e. dropped stars) of the sensor was neglected.

When the dropped star effect was introduced, the performance figure dropped to 94%. It was later discovered that the reason for this 'low' rate was due mostly to the limited size of the Yale Bright Star catalog at higher magnitudes (lower star brightness). In sparse regions of the sky, the pattern in the sensor FOV presented by the limited sky model occasionally consisted of only two or three dim stars. When one or two of them drop out because of the Star sensor magnitude precision limitations, at times, there was no pattern left to identify. Further experiments and parametric studies are under way using a more complete Harvard Smithsonian catalog.

The big gain was in terms of required memory. The serial algorithm stored over 70,000 star pairs at high precision in addition to code for a rather complex heuristic, artificial intelligence type of algorithm for a total size of 650 KBytes. The Neural algorithm used a connectionist data representation that was able to abstract from the star catalog, pattern class similarities, orthogonalities, and invariances in a highly compressed fashion. Network performance remained essentially constant until interconnect precision was decreased to less than four bits per synapse. 3000 synapses at four bits per synapse requires very little computer memory.

These simulation results were all derived from a monte carlo run of approximately 200,000 iterations using the simulator testbed.

## CONCLUSIONS

By means of a clever combination of several technologies and an appropriate data set representation, a star ID system using one of the most simple neural algorithms outperforms those using the classical serial ones in several aspects, even while running a software simulated neural network. The neural simulator is approximately ten times faster than the equivalent serial algorithm and requires less than one seventh the computer memory. With the transfer to neural VLSI technology, memory requirements will virtually disappear and processing speed will increase by at least an order of magnitude.

Where power and weight requirements scale with the hardware chip count, and every pound that must be launched into space costs millions of dollars, neural technology has enabled real-time on-board absolute attitude determination with no a priori information, that may eventually make several accessory satellite systems like horizon and sun sensors obsolete, while increasing the overall reliability of spacecraft systems.

### Acknowledgments

We would like to acknowledge many fruitfull conversations with C. E. Bell, J. Barhen and S. Gulati.

### References

R. W. H. van Bezooijen. Automated Star Pattern Recognition for Use With the Space Infrared Telescope Facility (SIRTIF). Paper for internal use at The Jet Propulsion Laboratory.

P. Gorman, T. J. Sejnowski. Workshop on Neural Network Devices and Applications (Jet Propulsion Laboratory, Pasaden, Ca.) Document D-4406, pp.224-237.

J. L. Lunkins. Star pattern Recognition for Real Time Attitude Determination. The Journal of Astronautical Science(1979).

D. E. Rummelhart, G. E. Hinton. Parallel Distributed Processing, eds. (MIT Press, Cambridge, Ma.) Vol. 1 pp. 318-364.

P. M. Salomon, T. A. Glavich. Image Signal Processing and Sub-Pixel Accuracy Star Trackers. SPIE vol. 252 Smart Sensors II (1980).

C.C.D. Image

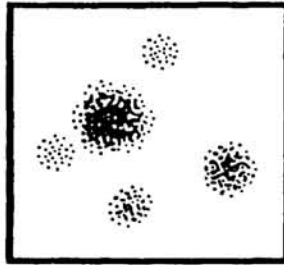

Preprocessor

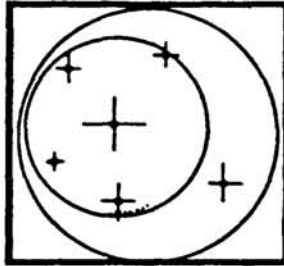

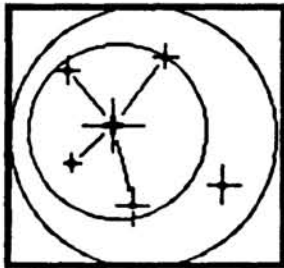

Distance
Map

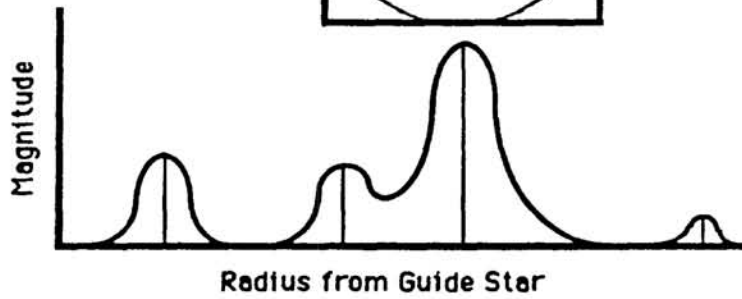

Neural
Sampler

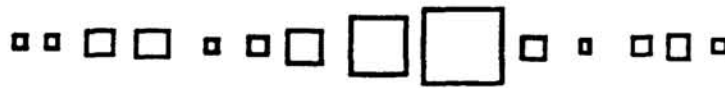

Neural
Output

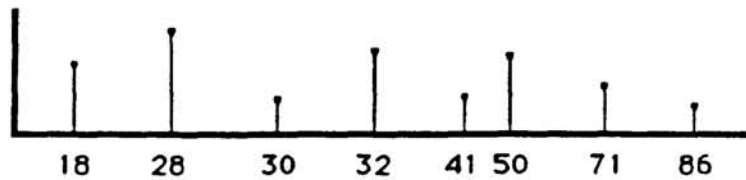

Star Attitude
Look-up Table

| St # | R.A. | Dec. |
|---|---|---|
| 27 | -1.3 | 2.45 |
| 28 | 1.8 | -1.1 |
| 29 | 0.2 | 0.68 |

PROTOTYPE HARDWARE IMPLEMENTATION

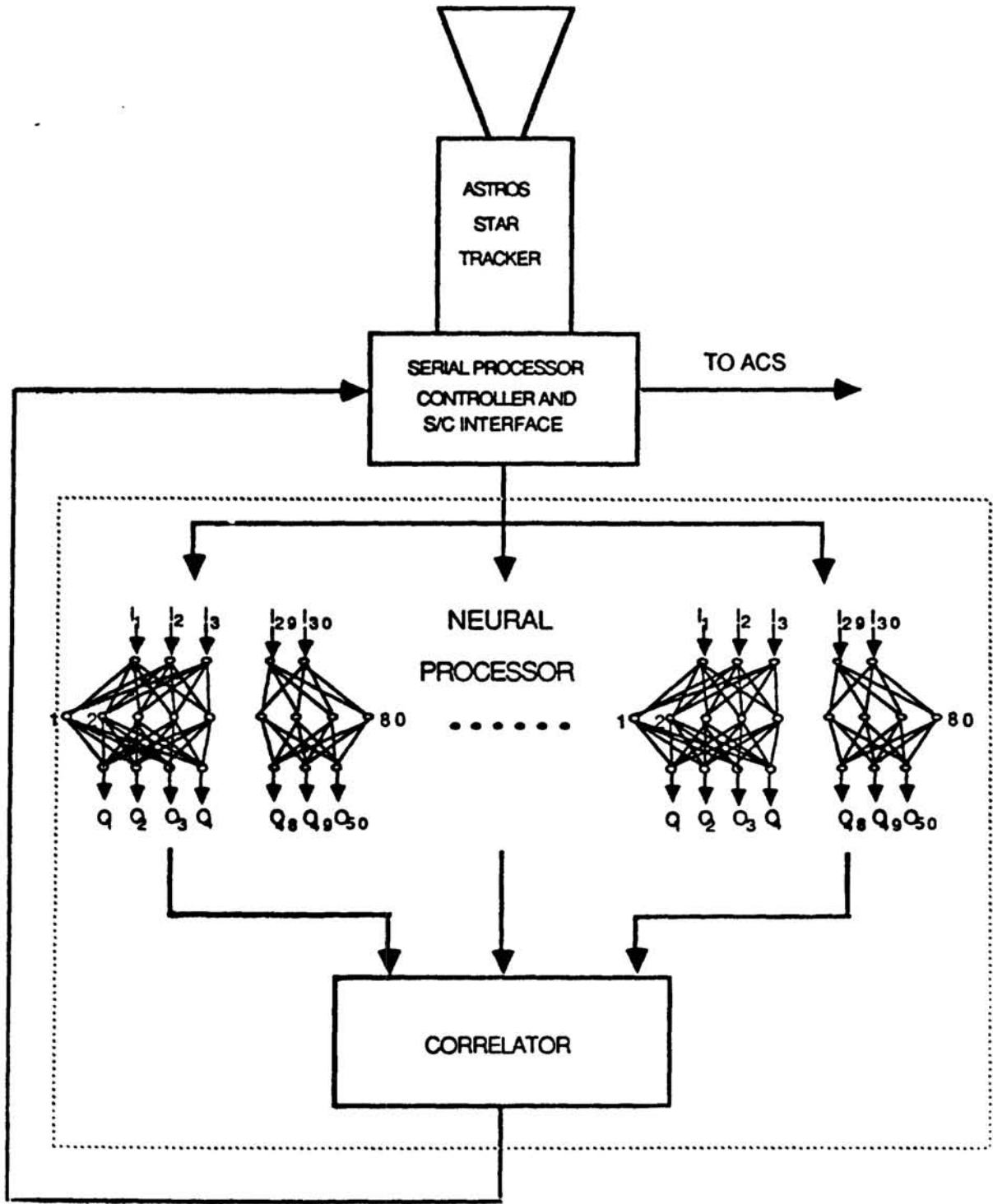